# Bayesian modeling of human concept learning

**Joshua B. Tenenbaum**
Department of Brain and Cognitive Sciences
Massachusetts Institute of Technology, Cambridge, MA 02139
jbt@psyche.mit.edu

## Abstract

I consider the problem of learning concepts from small numbers of positive examples, a feat which humans perform routinely but which computers are rarely capable of. Bridging machine learning and cognitive science perspectives, I present both theoretical analysis and an empirical study with human subjects for the simple task of learning concepts corresponding to axis-aligned rectangles in a multidimensional feature space. Existing learning models, when applied to this task, cannot explain how subjects generalize from only a few examples of the concept. I propose a principled Bayesian model based on the assumption that the examples are a random sample from the concept to be learned. The model gives precise fits to human behavior on this simple task and provides qualitative insights into more complex, realistic cases of concept learning.

## 1 Introduction

The ability to learn concepts from examples is one of the core capacities of human cognition. From a computational point of view, human concept learning is remarkable for the fact that very successful generalizations are often produced after experience with only a small number of positive examples of a concept (Feldman, 1997). While negative examples are no doubt useful to human learners in refining the boundaries of concepts, they are not necessary in order to make reasonable generalizations of word meanings, perceptual categories, and other natural concepts. In contrast, most machine learning algorithms require examples of both positive and negative instances of a concept in order to generalize at all, and many examples of both kinds in order to generalize successfully (Mitchell, 1997).

This paper attempts to close the gap between human and machine concept learning by developing a rigorous theory for concept learning from limited positive evidence and testing it against real behavioral data. I focus on a simple abstract task of interest to both cognitive science and machine learning: learning axis-parallel rectangles in $\Re^m$. We assume that each object $x$ in our world can be described by its values $(x_1, \ldots, x_m)$ on $m$ real-valued observable dimensions, and that each concept $C$ to be learned corresponds to a conjunction of independent intervals $(min_i(C) \leq x_i \leq max_i(C))$ along each dimension

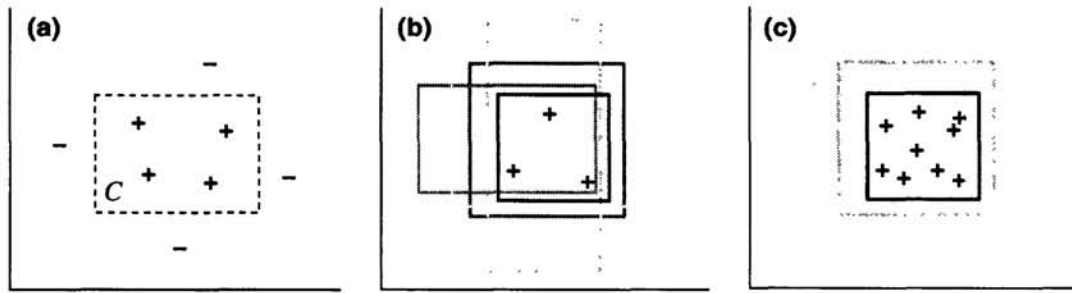

Figure 1: (a) A rectangle concept $C$. (b-c) The *size principle* in Bayesian concept learning: of the many hypotheses consistent with the observed positive examples, the smallest rapidly become more likely (indicated by darker lines) as more examples are observed.

*i*. For example, the objects might be people, the dimensions might be "cholesterol level" and "insulin level", and the concept might be "healthy levels". Suppose that "healthy levels" applies to any individual whose cholesterol and insulin levels are each greater than some minimum healthy level and less than some maximum healthy level. Then the concept "healthy levels" corresponds to a rectangle in the two-dimensional cholesterol/insulin space.

The problem of generalization in this setting is to infer, given a set of positive (+) and negative (-) examples of a concept $C$, which other points belong inside the rectangle corresponding to $C$ (Fig. 1a.). This paper considers the question most relevant for cognitive modeling: how to generalize from just a few positive examples?

In machine learning, the problem of learning rectangles is a common textbook example used to illustrate models of concept learning (Mitchell, 1997). It is also the focus of state-of-the-art theoretical work and applications (Dietterich et al., 1997). The rectangle learning task is not well known in cognitive psychology, but many studies have investigated human learning in similar tasks using simple concepts defined over two perceptually separable dimensions such as size and color (Shepard, 1987). Such impoverished tasks are worth our attention because they isolate the essential inductive challenge of concept learning in a form that is analytically tractable and amenable to empirical study in human subjects.

This paper consists of two main contributions. I first present a new theoretical analysis of the rectangle learning problem based on Bayesian inference and contrast this model's predictions with standard learning frameworks (Section 2). I then describe an experiment with human subjects on the rectangle task and show that, of the models considered, the Bayesian approach provides by far the best description of how people actually generalize on this task when given only limited positive evidence (Section 3). These results suggest an explanation for some aspects of the ubiquotous human ability to learn concepts from just a few positive examples.

## 2   Theoretical analysis

**Computational approaches to concept learning.** Depending on how they model a concept, different approaches to concept learning differ in their ability to generalize meaningfully from only limited positive evidence. *Discriminative* approaches embody *no* explicit model of a concept, but only a procedure for discriminating category members from members of mutually exclusive contrast categories. Most backprop-style neural networks and exemplar-based techniques (e.g. $K$-nearest neighbor classification) fall into this group, along with hybrid models like ALCOVE (Kruschke, 1992). These approaches are ruled out by definition; they cannot learn to discriminate positive and negative instances if they have seen only positive examples. *Distributional* approaches model a concept as a probability distribution over some feature space and classify new instances $x$ as members of $C$ if their

estimated probability $p(x|C)$ exceeds a threshold $\theta$. This group includes "novelty detection" techniques based on Bayesian nets (Jaakkola et al., 1996) and, loosely, autoencoder networks (Japkowicz et al., 1995). While $p(x|C)$ can be estimated from only positive examples, novelty detection also requires negative examples for principled generalization, in order to set an appropriate threshold $\theta$ which may vary over many orders of magnitude for different concepts. For learning from positive evidence only, our best hope are algorithms that treat a new concept $C$ as an *unknown subset* of the universe of objects and decide how to generalize $C$ by finding "good" subsets in a hypothesis space $H$ of possible concepts.

**The Bayesian framework.** For this task, the natural hypothesis space $H$ corresponds to all rectangles in the plane. The central challenge in generalizing using the subset approach is that any small set of examples will typically be consistent with many hypotheses (Fig. 1b). This problem is not unique to learning rectangles, but is a universal dilemma when trying to generalize concepts from only limited positive data. The Bayesian solution is to embed the hypothesis space in a probabilistic model of our observations, which allows us to weight different consistent hypotheses as more or less likely to be the true concept based on the particular examples observed. Specifically, we assume that the examples are generated by *random sampling* from the true concept. This leads to the *size principle*: smaller hypotheses become more likely than larger hypotheses (Fig. 1b – darker rectangles are more likely), and they become exponentially more likely as the number of consistent examples increases (Fig. 1c). The size principle is the key to understanding how we can learn concepts from only a few positive examples.

**Formal treatment.** We observe $n$ positive examples $X = \{x^{(1)}, \ldots, x^{(n)}\}$ of concept $C$ and want to compute the *generalization function* $p(y \in C|X)$, i.e. the probability that some new object $y$ belongs to $C$ given the observations $X$. Let each rectangle hypothesis $h$ be denoted by a quadruple $(l_1, l_2, s_1, s_2)$, where $l_i \in [-\infty, \infty]$ is the location of $h$'s lower-left corner and $s_i \in [0, \infty]$ is the size of $h$ along dimension $i$.

Our probabilistic model consists of a prior density $p(h)$ and a likelihood function $p(X|h)$ for each hypothesis $h \in H$. The likelihood is determined by our assumption of randomly sampled positive examples. In the simplest case, each example in $X$ is assumed to be independently sampled from a uniform density over the concept $C$. For $n$ examples we then have:

$$
\begin{aligned}
p(X|h) &= 1/|h|^n \text{ if } \forall j, x^{(j)} \in h \\
&= 0 \text{ otherwise,}
\end{aligned}
\tag{1}
$$

where $|h|$ denotes the size of $h$. For rectangle $(l_1, l_2, s_1, s_2)$, $|h|$ is simply $s_1 s_2$. Note that because each hypothesis must distribute one unit mass of likelihood over its volume for each example ($\int_{x \in h} p(x|h)dh = 1$), the probability density for smaller consistent hypotheses is greater than for larger hypotheses, and exponentially greater as a function of $n$. Figs. 1b,c illustrate this size principle for scoring hypotheses (darker rectangles are more likely).

The appropriate choice of $p(h)$ depends on our background knowledge. If we have no *a priori* reason to prefer any rectangle hypothesis over any other, we can choose the scale- and location-invariant *uninformative* prior, $p(h) = p(l_1, l_2, s_1, s_2) = 1/(s_1, s_2)$. In any realistic application, however, we will have some prior information. For example, we may know the expected size $\sigma_i$ of rectangle concepts along dimension $i$ in our domain, and then use the associated maximum entropy prior $p(l_1, l_2, s_1, s_2) = \exp\{-(s_1/\sigma_1 + s_2/\sigma_2)\}$.

The generalization function $p(y \in C|X)$ is computed by integrating the predictions of all hypotheses, weighted by their posterior probabilities $p(h|X)$:

$$
p(y \in C|X) = \int_{h \in H} p(y \in C|h)\, p(h|X)\, dh,
\tag{2}
$$

where from Bayes' theorem $p(h|X) \propto p(X|h)p(h)$ (normalized such that $\int_{h \in H} p(h|X)dh = 1$), and $p(y \in C|h) = 1$ if $y \in h$ and 0 otherwise. Under the

uninformative prior, this becomes:

$$p(y \in C|X) = \left[ \frac{1}{(1 + \tilde{d}_1/r_1)(1 + \tilde{d}_2/r_2)} \right]^{n-1}. \tag{3}$$

Here $r_i$ is the maximum distance between the examples in $X$ along dimension $i$, and $\tilde{d}_i$ equals 0 if $y$ falls inside the range of values spanned by $X$ along dimension $i$, and otherwise equals the distance from $y$ to the nearest example in $X$ along dimension $i$. Under the expected-size prior, $p(y \in C|X)$ has no closed form solution valid for all $n$. However, except for very small values of $n$ (e.g. $< 3$) and $r_i$ (e.g. $< \sigma_i/10$), the following approximation holds to within 10% (and usually much less) error:

$$p(y \in C|X) \approx \frac{\exp\{-(\tilde{d}_1/\sigma_1 + \tilde{d}_2/\sigma_2)\}}{[(1 + \tilde{d}_1/r_1)(1 + \tilde{d}_2/r_2)]^{n-1}}. \tag{4}$$

Fig. 2 (left column) illustrates the Bayesian learner's contours of equal probability of generalization (at $p = 0.1$ intervals), for different values of $n$ and $r_i$. The bold curve corresponds to $p(y \in C|X) = 0.5$, a natural boundary for generalizing the concept. Integrating over all hypotheses weighted by their size-based probabilities yields a broad gradient of generalization for small $n$ (row 1) that rapidly sharpens up to the smallest consistent hypothesis as $n$ increases (rows 2-3), and that extends further along the dimension with a broader range $r_i$ of observations. This figure reflects an expected-size prior with $\sigma_1 = \sigma_2 = axis\_width/2$; using an uninformative prior produces a qualitatively similar plot.

**Related work: MIN and Weak Bayes.** Two existing subset approaches to concept learning can be seen as variants of this Bayesian framework. The classic MIN algorithm generalizes no further than the smallest hypothesis in $H$ that includes all the positive examples (Bruner et al., 1956; Feldman, 1997). MIN is a PAC learning algorithm for the rectangles task, and also corresponds to the maximum likelihood estimate in the Bayesian framework (Mitchell, 1997). However, while it converges to the true concept as $n$ becomes large (Fig. 2, row 3), it appears extremely conservative in generalizing from very limited data (Fig. 2, row 1).

An earlier approach to Bayesian concept learning, developed independently in cognitive psychology (Shepard, 1987) and machine learning (Haussler et al., 1994; Mitchell, 1997), was an important inspiration for the framework of this paper. I call the earlier approach *weak Bayes*, because it embodies a different generative model that leads to a much weaker likelihood function than Eq. 1. While Eq. 1 came from assuming examples sampled randomly from the true concept, weak Bayes assumes the examples are generated by an arbitrary process *independent* of the true concept. As a result, the size principle for scoring hypotheses does not apply; all hypotheses consistent with the examples receive a likelihood of 1, instead of the factor of $1/|h|^n$ in Eq. 1. The extent of generalization is then determined solely by the prior; for example, under the expected-size prior,

$$p(y \in C|X) = \exp\{-(\tilde{d}_1/\sigma_1 + \tilde{d}_2/\sigma_2)\}. \tag{5}$$

Weak Bayes, unlike MIN, generalizes reasonably from just a few examples (Fig. 2, row 1). However, because Eq. 5 is independent of $n$ or $r_i$, weak Bayes does not converge to the true concept as the number of examples increases (Fig. 2, rows 2-3), nor does it generalize further along axes of greater variability. While weak Bayes is a natural model when the examples really are generated independently of the concept (e.g. when the learner himself or a random process chooses objects to be labeled "positive" or "negative" by a teacher), it is clearly limited as a model of learning from deliberately provided positive examples.

In sum, previous subset approaches each appear to capture a different aspect of how humans generalize concepts from positive examples. The broad similarity gradients that emerge

from weak Bayes seem most applicable when only a few broadly spaced examples have been observed (Fig. 2, row 1), while the sharp boundaries of the MIN rule appear more reasonable as the number of examples increases or their range narrows (Fig. 2, rows 2-3). In contrast, the Bayesian framework guided by the size principle automatically interpolates between these two regimes of similarity-based and rule-based generalization, offering the best hope for a complete model of human concept learning.

## 3   Experimental data from human subjects

This section presents empirical evidence that our Bayesian model – but neither MIN nor weak Bayes – can explain human behavior on the simple rectangle learning task. Subjects were given the task of guessing 2-dimensional rectangular concepts from positive examples only, under the cover story of learning about the range of healthy levels of insulin and cholesterol, as described in Section 1. On each trial of the experiment, several dots appeared on a blank computer screen. Subjects were told that these dots were randomly chosen examples from some arbitrary rectangle of "healthy levels," and their job was to guess that rectangle as nearly as possible by clicking on-screen with the mouse. The dots were in fact randomly generated on each trial, subject to the constraints of three independent variables that were systematically varied across trials in a $(6 \times 6 \times 6)$ factorial design. The three independent variables were the horizontal range spanned by the dots (.25, .5, 1, 2, 4, 8 units in a 24-unit-wide window), vertical range spanned by the dots (same), and number of dots (2, 3, 4, 6, 10, 50). Subjects thus completed 216 trials in random order. To ensure that subjects understood the task, they first completed 24 practice trials in which they were shown, after entering their guess, the "true" rectangle that the dots were drawn from. [1]

The data from 6 subjects is shown in Fig. 3a, averaged across subjects and across the two directions (horizontal and vertical). The extent $d$ of subjects' rectangles beyond $r$, the range spanned by the observed examples, is plotted as a function of $r$ and $n$, the number of examples. Two patterns of generalization are apparent. First, $d$ increases monotonically with $r$ and decreases with $n$. Second, the rate of increase of $d$ as a function of $r$ is much slower for larger values of $n$.

Fig. 3b shows that neither MIN nor weak Bayes can explain these patterns. MIN always predicts zero generalization beyond the examples – a horizontal line at $d = 0$ – for all values of $r$ and $n$. The predictions of weak Bayes are also independent of $r$ and $n$: $d = \sigma \log 2$, assuming subjects give the tightest rectangle enclosing all points $y$ with $p(y \in C|X) > 0.5$.

Under the same assumption, Figs. 3c,d show our Bayesian model's predicted bounds on generalization using uninformative and expected-size priors, respectively. Both versions of the model capture the qualitative dependence of $d$ on $r$ and $n$, confirming the importance of the size principle in guiding generalization independent of the choice of prior. However, the uninformative prior misses the nonlinear dependence on $r$ for small $n$, because it assumes an ideal scale invariance that clearly does not hold in this experiment (due to the fixed size of the computer window in which the rectangles appeared). In contrast, the expected-size prior naturally embodies prior knowledge about typical scale in its one free parameter $\sigma$. A reasonable value of $\sigma = 5$ units (out of the 24-unit-wide window) yields an excellent fit to subjects' average generalization behavior on this task.

## 4   Conclusions

In developing a model of concept learning that is at once computationally principled and able to fit human behavior precisely, I hope to have shed some light on how people are able

to infer the correct extent of a concept from only a few positive examples. The Bayesian model has two key components: (1) a generalization function that results from integrating the predictions of all hypotheses weighted by their posterior probability; (2) the assumption that examples are sampled from the concept to be learned, and not independently of the concept as previous weak Bayes models have assumed. Integrating predictions over the whole hypothesis space explains why either broad gradients of generalization (Fig. 2, row 1) or sharp, rule-based generalization (Fig. 2, row 3) may emerge, depending on how peaked the posterior is. Assuming examples drawn randomly from the concept explains why learners do not weight all consistent hypotheses equally, but instead weight more specific hypotheses higher than more general ones by a factor that increases exponentially with the number of examples observed (the *size principle*).

This work is being extended in a number of directions. Negative instances, when encountered, are easily accomodated by assigning zero likelihood to any hypotheses containing them. The Bayesian formulation applies not only to learning rectangles, but to learning concepts in any measurable hypothesis space – wherever the size principle for scoring hypotheses may be applied. In Tenenbaum (1999), I show that the same principles enable learning number concepts and words for kinds of objects from only a few positive examples. [2] I also show how the size principle supports much more powerful inferences than this short paper could demonstrate: automatically detecting incorrectly labeled examples, selecting relevant features, and determining the complexity of the hypothesis space. Such inferences are likely to be necessary for learning in the complex natural settings we are ultimately interested in.

## Acknowledgments

Thanks to M. Bernstein, W. Freeman, S. Ghaznavi, W. Richards, R. Shepard, and Y. Weiss for helpful discussions. The author was a Howard Hughes Medical Institute Predoctoral Fellow.

## Footnotes

[1] Because dots were drawn randomly, the "true" rectangles that subjects saw during practice were quite variable and were rarely the "correct" response according to *any* theory considered here. Thus it is unlikely that this short practice was responsible for any consistent trends in subjects' behavior.

[2]In the framework of inductive logic programming, Muggleton (preprint) has independently proposed that similar principles may allow linguistic grammars to be learned from positive data only.

## References

Bruner, J. A., Goodnow, J. S., & Austin, G. J. (1956). *A study of thinking*. New York: Wiley.

Dietterich, T., Lathrop, R., & Lozano-Perez, T. (1997). Solving the multiple-instance problem with axis-parallel rectangles. *Artificial Intelligence* **89**(1-2), 31-71.

Feldman, J. (1997). The structure of perceptual categories. *J. Math. Psych.* **41**, 145-170.

Haussler, D., Kearns, M., & Schapire, R. (1994). Bounds on the sample complexity of Bayesian learning using information theory and the VC-dimension. *Machine Learning* **14**, 83-113.

Jaakkola, T., Saul, L., & Jordan, M. (1996) Fast learning by bounding likelihoods in sigmoid type belief networks. *Advances in Neural Information Processing Systems 8*.

Japkowicz, N., Myers, C., & Gluck, M. (1995). A novelty detection approach to classification. *Proceedings of the 14th International Joint Conference on Aritifical Intelligence*.

Kruschke, J. (1992). ALCOVE: An exemplar-based connectionist model of category learning. *Psych. Rev.* **99**, 22-44.

Mitchell, T. (1997). *Machine Learning*. McGraw-Hill.

Muggleton, S. (preprint). Learning from positive data. Submitted to *Machine Learning*.

Shepard, R. (1987). Towards a universal law of generalization for psychological science. *Science* **237**, 1317-1323.

Tenenbaum, J. B. (1999). *A Bayesian Framework for Concept Learning*. Ph. D. Thesis, MIT Department of Brain and Cognitive Sciences.

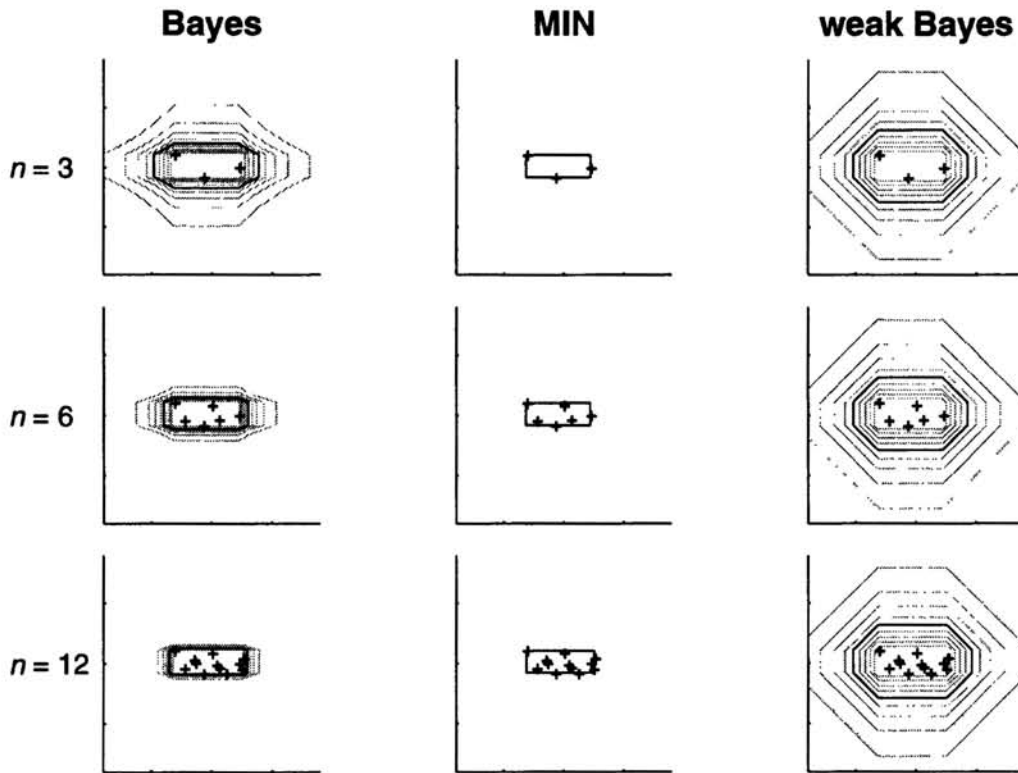

Figure 2: Performance of three concept learning algorithms on the rectangle task.

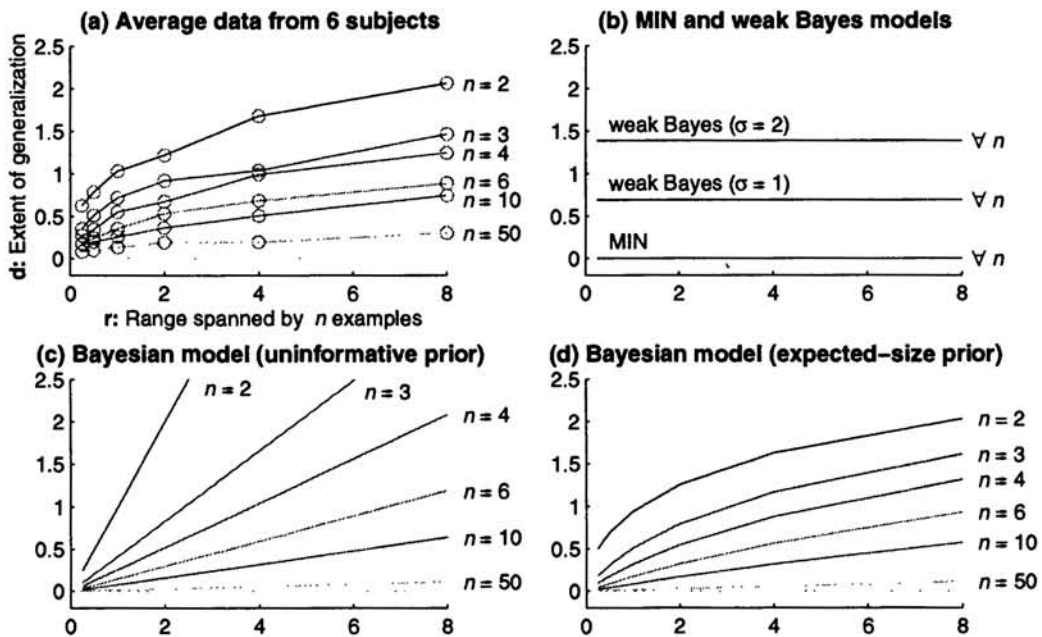

Figure 3: Data from human subjects and model predictions for the rectangle task.
